# CPR for CSPs: A Probabilistic Relaxation of Constraint Propagation

**Luis E. Ortiz**
ECE Dept, Univ. of Puerto Rico, Mayagüez, PR 00681-9042
`leortiz@ece.uprm.edu`

## Abstract

This paper proposes *constraint propagation relaxation (CPR)*, a probabilistic approach to classical constraint propagation that provides another view on the whole parametric family of survey propagation algorithms $\mathrm{SP}(\rho)$. More importantly, the approach elucidates the implicit, but fundamental assumptions underlying $\mathrm{SP}(\rho)$, thus shedding some light on its effectiveness and leading to applications beyond $k$-SAT.

## 1 Introduction

*Survey propagation (SP)* is an algorithm for solving $k$-SAT recently developed in the physics community [1, 2] that exhibits excellent empirical performance on "hard" instances. To understand the behavior of SP and its effectiveness, recent work (see Maneva et al. [3] and the references therein) has concentrated on establishing connections to *belief propagation (BP)* [4], a well-known approximation method for computing posterior probabilities in probabilistic graphical models. Instead, this paper argues that it is perhaps more natural to establish connections to *constraint propagation (CP)*, another message-passing algorithm tailored to *constraint satisfaction problems (CSPs)* that is well-known in the AI community. The ideas behind CP were first proposed by Waltz [5] [1] Yet, CP has received considerably less attention than BP lately.

This paper reconnects BP to CP in the context of CSPs by proposing a probabilistic relaxation of CP that generalizes it. Through the approach, it is easy to see the exact, implicit underlying assumptions behind the entire family of survey propagation algorithms $\mathrm{SP}(\rho)$. (Here, the approach is presented in the context of $k$-SAT; it will be described in full generality in a separate document.) In short, the main point of this paper is that *survey propagation algorithms are instances of a natural generalization of constraint propagation and have simple interpretations in that context.*

## 2 Constraint Networks and Propagation

This section presents a brief introduction to the graphical representation of CSPs and CP, and concentrates on the aspects that are relevant to this paper. [2]

A *constraint network (CN)* is the graphical model for CSPs used in the AI community. Of interest here is the CN based on the *hidden transformation*. (See Bacchus et al. [9] for more information on the different transformations and their properties.) It has a bipartite graph where every variable and constraint is each represented by a node or vertex in the graph and there is an edge between a variable $i$ and a constraint $a$ if and only if $a$ is a function of $i$ (see figure 1). From now on, a CN with a tree graph is referred to as a *tree CN*, and a CN with an arbitrary graph as an *arbitrary CN*.

Constraint propagation is typically used as part of a depth-first search algorithm for solving CSPs. The search algorithm works by extending partial assignments, usually one variable at a time, during the search. The algorithm is called *backtracking search* because one can backtrack and change the value of a previously assigned variable when the search reaches an illegal assignment.

CP is often applied either as a preprocessing step or after an assignment to a variable is made. The objective is to reduce the domains of the variables by making them locally consistent with the current partial assignment. The propagation process starts with the belief that for every value assignment $v_i$ in the domain of each variable $i$ there exists a solution with $v_i$ assigned to $i$. The process then attempts to correct this *a priori* belief by locally propagating constraint information. It is well-known that CP, unlike BP, *always* converges, regardless of the structure of the CN graph. This is because no possible solution is ignored at the start and none ever removed during the process. In the end, CP produces potentially reduced variable domains that are in fact locally consistent. In turn, the

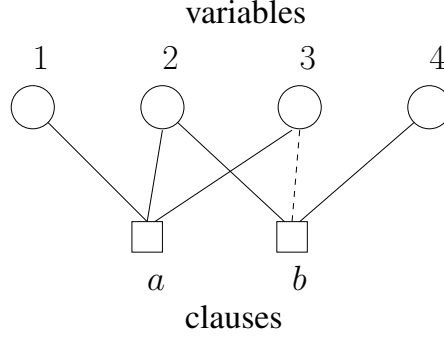

variables

clauses

Figure 1: The graph of the constraint network corresponding to the 3-SAT formula $f(x) = (x_1 \lor x_2 \lor x_3) \land (x_2 \lor \bar{x}_3 \lor x_4)$, which has four variables and two clauses; the first and second clause are denoted in the figure by $a$ and $b$, respectively. Following the convention of the SP community, clause and variable nodes are drawn as boxes and circles, respectively; also, if a variable appears as a negative literal in a clause (*e.g.*, variable 3 in clause $b$), the edge between them is drawn as a dashed line.

resulting search space is at worst no larger than the original but potentially smaller while still containing all possible solutions. The computational efficiency and effectiveness of CP in practice has made it a popular algorithm in the CSP community.

## 3 Terminology and Notation

Let $V(a)$ be the set of variables that appear in constraint $a$ and $C(i)$ the set of constraints in which variable $i$ appears. Let also $V_i(a) \equiv V(a) - \{i\}$ and $C_a(i) \equiv C(i) - \{a\}$. In $k$-SAT, the constraints are the clauses, each variable is binary, with domain $\{0, 1\}$, and a solution corresponds to a satisfying assignment. If $i \in V(a)$, denote by $s_{a,i}$ the value assignment to variable $i$ that guarantees the satisfiability of clause $a$; and denote the other possible assignment to $i$ by $u_{a,i}$. Finally, let $C_a^s(i)$ and $C_a^u(i)$ be the set of clauses in $C_a(i)$ where variable $i$ appears in the *same* and *different* literal form as it does in clause $a$, respectively.

The $k$-SAT formula under consideration is denoted by $f$. It is convenient to introduce notation for formulae associated to the CN that results from removing variables or constraints from $f$. Let $f_a$ be the function that results from removing clause $a$ from $f$ (see figure 2), and

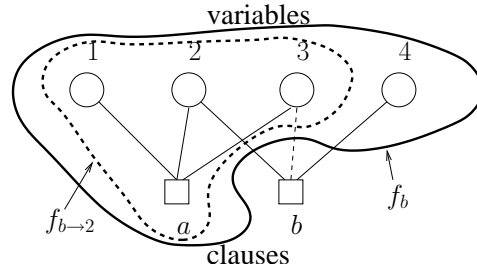

variables

clauses

Figure 2: The graph inside the continuous curve is the CN graph for the formula $f_b$ that results from removing clause $b$ from $f$. The graph inside the dashed curve is the CN graph for $f_{b\rightarrow 2}$, which corresponds to the formula for the connected component of the CN graph for $f_b$ that contains variable 2.

similarly, abusing notation, let $f_i$ be the function that results from removing variable $i$ from $f$. Let $f_{a\rightarrow i}$ be the function that corresponds to the connected component of the CN graph for $f_a$ that contains variable $i \in V(a)$, and let $f_{i\rightarrow a}$ be the function that corresponds to the connected component of the CN graph for $f_i$ that contains $a \in C(i)$. (Naturally, if node $a$ is not a separator of the CN graph for $f$, $f_a$ has a single connected component, which leads to $f_{a\rightarrow i} = f_a$; similarly for $f_i$.)

It is convenient to use a simple, if perhaps unusual, representation of sets in order to track the domains of the variables during the propagation process. Each subset $A$ of a set $S$ of size $m$ is represented as a bit array of $m$ elements where component $k$ in the array is set to 1 if $k$ is in $A$ and to 0 otherwise. For instance, if $S = \{0, 1\}$, then the array $[00]$ represents $\emptyset$, and similarly, $[01]$, $[10]$ and $[11]$ represent $\{0\}$, $\{1\}$ and $\{0, 1\}$, respectively.

It is also useful to introduce the concept of *(globally) consistent domains* of variables and SAT functions. Let $\mathcal{S}_f = \{x | x \text{ satisfies } f\}$ be the set of assignments that satisfy $f$. Given a complete assignment $x$, denote by $x_{-i}$ the assignments to all the variables except $i$; thus, $x = (x_1, \ldots, x_n) = (x_i, x_{-i})$. Let the set $\mathcal{W}_i$ be the *consistent domain of variable $i$ in $f$* if $\mathcal{W}_i = \{x_i | x = (x_i, x_{-i}) \in \mathcal{S}_f \text{ for some } x_{-i}\}$; that is, $\mathcal{W}_i$ contains the set of all possible values that variable $i$ can take in an assignment that satisfies $f$. Let the set $\mathcal{W}$ be the *consistent domain of $f$* if $\mathcal{W} = \times_{i=1}^{n} \mathcal{W}_i$ and, for all $i$, $\mathcal{W}_i$ is the consistent domain of variable $i$ in $f$.

Finally, some additional terminology classifies variables of a SAT function given a satisfying assignment. Given a function $f$ and a satisfying assignment $x$, let variable $i$ be *fixed* if changing only its assignment $x_i$ in $x$ does not produce another satisfying assignment for $f$; and be *free* otherwise.

## 4 Propagation Algorithms for Satisfiability

**Constraint Propagation.** In CP for $k$-SAT, the message $M_{a \to i}$ that clause $a$ sends to variable $i$ is an array of binary values indexed by the elements of the domain of $i$; similarly, for the message $M_{i \to a}$ that variable $i$ sends to clause $a$. Intuitively, for all $x_i \in \{0, 1\}$, $M_{i \to a}(x_i) = 1$ if and only if assigning value $x_i$ to variable $i$ is "ok" with all clauses other than $a$. Formally, $M_{i \to a}(x_i) = 1$ if and only if $f_{a \to i}$ has a satisfying assignment with $x_i$ assigned to variable $i$ (or in other words, $x_i$ is in the consistent domain of $i$ in $f_{a \to i}$). Similarly, $M_{a \to i}(x_i) = 1$ if and only if clause $a$ is "ok" with assigning value $x_i$ to variable $i$; or formally, $M_{a \to i}(x_i) = 1$ if and only if $f_{i \to a}$ has a satisfying assignment with $x_i$ assigned to variable $i$, or assigning $x_i$ to variable $i$ by itself satisfies $a$. It is convenient to denote $M_{i \to a}(x_i)$ and $M_{i \to a}(x_i)$ by $M_{a \to i}^{x_i}$ and $M_{a \to i}^{x_i}$, respectively. In addition, $M_{i \to a}^{s_{a,i}}$, $M_{i \to a}^{u_{a,i}}$, $M_{a \to i}^{s_{a,i}}$ and $M_{a \to i}^{u_{a,i}}$ are simply denoted by $M_{i \to a}^{s}$, $M_{i \to a}^{u}$, $M_{a \to i}^{s}$ and $M_{a \to i}^{u}$, respectively.

In summary, we can write CP for $k$-SAT as follows.

- *Messages that clause $a$ sends to variable $i$:*

$$M_{a \to i}^{x_i} = 1 \text{ if and only if } x_i = s_{a,i} \text{ or, there exists } j \in V_i(a), \text{ s.t. } M_{j \to a}^{s} = 1. \quad (1)$$

- *Messages that variable $i$ sends to clause $a$:*

$$M_{i \to a}^{x_i} = 1 \text{ if and only if for all } b \in C_a(i), M_{b \to i}^{x_i} = 1. \quad (2)$$

It is convenient to express CP mathematically as follows.

- *Messages that clause $a$ sends to variable $i$:*

$$M_{a \to i}^{x_i} = \begin{cases} 1, & \text{if } x_i = s_{a,i}, \\ 1 - \prod_{j \in V_i(a)}(1 - M_{j \to a}^{s}), & \text{if } x_i = u_{a,i}. \end{cases}$$

- *Messages that variable $i$ sends to clause $a$:* $M_{i \to a}^{x_i} = \prod_{b \in C_a(i)} M_{b \to i}^{x_i}$.

In order to guarantee convergence, the message values in CP are initialized as $M_{i \to a}^{s} = 1$, $M_{i \to a}^{u} = 1$, $M_{a \to i}^{u} = 1$, and naturally, $M_{a \to i}^{s} = 1$. This initialization encodes the *a priori* belief that every assignment is a solution. CP attempts to "correct" or update this belief through the local propagation of constraint information. In fact, the expressions in CP force the messages to be *locally* consistent. By being initially conservative about the consistent domains, no satisfying assignment is discarded during the propagation process.

Once CP converges, for each variable $i$, its *locally-consistent domain* becomes $\{x_i | \prod_{a \in C(i)} M_{a \to i}^{x_i} = 1\} = \{x_i | \prod_{a \in C(i): x_i = u_{a,i}} M_{a \to i}^{u} = 1\} \in 2^{\{0,1\}}$. For general CSPs, CP is usually very effective because it can significantly reduce the original domain of the variables,

leading to a smaller search space of possible assignments. It should be noted that in the particular case of $k$-SAT with arbitrary CNs, CP is usually only effective after some variables have already being assigned during the search, because those (partial) assignments can lead to "boundary conditions." Without such boundary conditions, however, CP never reduces the domain of the variables in $k$-SAT, as can be easily seen from the expressions above.

On the other hand, when CP is applied to tree CNs, it exhibits additional special properties. For example, convergence is actually guaranteed *regardless* of how the messages are initialized, because of the boundary conditions imposed by the leaves of the tree. Also, the final messages are in fact *globally* consistent (*i.e.*, *all* the messages are consistent with their definition). Therefore, the locally-consistent domains are in fact *the* consistent domains. Whether the formula is satisfiable, or not, can be determined immediately after applying CP. If the formula is not satisfiable, the consistent domains will be empty sets. If the formula is in fact satisfiable, applying depth-first search always finds a satisfying assignment without the need to backtrack.

We can express CP in a way that looks closer to SP and BP. Using the reparametrization $\Gamma_{a \to i} = 1 - M_{a \to i}^u$, we get the following expression of CP.

- *Message that clause $a$ sends to variable $i$:* $\Gamma_{a \to i} = \prod_{j \in V_i(a)} (1 - M_{j \to a}^s)$.
- *Message that variable $i$ sends to clause $a$:* $M_{i \to a}^s = \prod_{b \in C_a^u(i)} (1 - \Gamma_{b \to i})$.

**Survey Propagation.**   Survey propagation has become a very popular propagation algorithm for $k$-SAT. It was developed in the physics community by Mézard et al. [2]. The excitement around SP comes from its excellent empirical performance on hard satisfiability problems; that is, $k$-SAT formulae with a ratio $\alpha$ of the number of clauses to the number of variables near the so called *satisfiability threshold* $\alpha_c$.

The following is a description of an SP-inspired family of message-passing procedures, parametrized by $\rho \in [0,1]$. It is often denoted by SP($\rho$), and contains BP ($\rho = 0$) and (pure) SP ($\rho = 1$).

- *Message that clause $a$ sends to variable $i$:*

$$\eta_{a \to i} = \prod_{j \in V_i(a)} \frac{\Pi_{j \to a}^u}{\Pi_{j \to a}^u + \Pi_{j \to a}^s + \Pi_{j \to a}^*}$$

- *Messages that variable $i$ sends to clause $a$:*

$$
\begin{aligned}
\Pi_{i \to a}^u &= \left(1 - \rho \prod_{b \in C_a^u(i)} (1 - \eta_{b \to i})\right) \prod_{b \in C_a^s(i)} (1 - \eta_{b \to i}) \\
\Pi_{i \to a}^s &= \prod_{b \in C_a^u(i)} (1 - \eta_{b \to i}) \left(1 - \prod_{b \in C_a^s(i)} (1 - \eta_{b \to i})\right) \\
\Pi_{i \to a}^* &= \prod_{b \in C_a^u(i)} (1 - \eta_{b \to i}) \prod_{b \in C_a^s(i)} (1 - \eta_{b \to i}) = \prod_{b \in C_a(i)} (1 - \eta_{b \to i})
\end{aligned}
$$

SP was originally derived via arguments and concepts from physics. A simple derivation based on a probabilistic interpretation of CP is given in the next section of the paper. The derivation presented here elucidates the assumptions that SP algorithms make about the satisfiability properties and structure of $k$-SAT formulae. However, it is easy to establish strong equivalence relations between the different propagation algorithms even at the basic level, before introducing the probabilistic interpretation (details omitted).

## 5   A Probabilistic Relaxation of Constraint Propagation for Satisfiability

The main idea behind *constraint propagation relaxation (CPR)* is to introduce a probabilistic model for the $k$-SAT formula and view the messages as random variables in that model. If the formula $f$ has $n$ variables, the sample space $\Omega = (2^{\{0,1\}})^n$ is the set of the $n$-tuple whose components are *subsets* of the set of possible values that each variable $i$ can take (*i.e.*, subsets of $\{0,1\}$). The "true probability law" $\mathbf{P}_f$ of a SAT formula $f$ that corresponds to CP is defined in terms of the consistent domain of $f$: for all $\mathcal{W} \in \Omega$,

$$\mathbf{P}_f(\mathcal{W}) = \begin{cases} 1, & \text{if } \mathcal{W} \text{ is the consistent domain of } f, \\ 0, & \text{otherwise.} \end{cases}$$

Clearly, if we could compute the consistent domains of the remaining variables after each variable assignment during the search, there would be no need to backtrack. But, while it is easy to compute consistent domains for tree CNs, it is actually hard in general for arbitrary CNs. Thus, it is generally hard to compute $\mathbf{P}_f$. (CNs with graphs of bounded tree-width are a notable exception.)

However, the probabilistic interpretation will allow us to introduce "bias" on $\Omega$, which leads to a heuristic for dynamically ordering both the variables and their values during search. As shown in this section, it turns out that for arbitrary CNs, survey propagation algorithms attempt to compute different "approximations" or "relaxations" of $\mathbf{P}_f$ by making different assumptions about its "probabilistic structure."

Let us now view each message $M^s_{a \to i}$, $M^u_{a \to i}$, $M^s_{i \to a}$, and $M^u_{i \to a}$ for each variable $i$ and clause $a$ as a (Bernoulli) random variable in some probabilistic model with sample space $\Omega$ and a, now *arbitrary*, probability law $\mathbf{P}$. [3] Formally, for each clause $a$, variable $i$ and possible assignment value $x_i \in \{0, 1\}$, we define

$$M^{x_i}_{a \to i} \sim \text{Bernoulli}(p^{x_i}_{a \to i}) \text{ and } M^{x_i}_{i \to a} \sim \text{Bernoulli}(p^{x_i}_{i \to a})$$

where $p^{x_i}_{a \to i} = \mathbf{P}(M^{x_i}_{a \to i} = 1)$ and $p^{x_i}_{i \to a} = \mathbf{P}(M^{x_i}_{i \to a} = 1)$. This is a distribution over *all* possible subsets (*i.e.*, the power set) of the domain of each variable, *not* just over the variable's domain itself. Also, clearly we do not need to worry about $p^s_{a \to i}$ because it is always 1, by the definition of $M^s_{a \to i}$.

The following is a description of how we can use those probabilities during search. In the SP community, the resulting heuristic search is called "decimation" [1, 2]. If we believe that $\mathbf{P}$ "closely approximates" $\mathbf{P}_f$, and know the probability $p^{x_i}_i \equiv \mathbf{P}(M^{x_i}_{a \to i} = 1$ for all $a \in C(i))$ that $x_i$ is in the consistent domain for variable $i$ of $f$, for every variable $i$, clause $a$ and possible assignment $x_i$, we can use them to dynamically order both the variables and the values they can take during search. Specifically, we first compute $p^1_i = \mathbf{P}(M^u_{a \to i} = 1$ for all $a \in C^-(i))$ and $p^0_i = \mathbf{P}(M^u_{a \to i} = 1$ for all $a \in C^+(i))$ for each variable $i$, where $C^+(i)$ and $C^-(i)$ are the sets of clauses where variable $i$ appears as a *positive* and a *negative* literal, respectively. Using those probability values, we then compute what the SP community calls the "bias" of $i$: $|p^1_i - p^0_i|$. The variable to assign next is the one with the largest bias. [4] We would set that variable to the value of largest probability; for instance, if variable $i$ has the largest bias, then we set $i$ next, to 1 if $p^1_i > p^0_i$, and to 0 if $p^1_i < p^0_i$. The objective is then to compute or estimate those probabilities.

The following are (independence) assumptions about the random variables (*i.e.*, messages) used in this section. The assumptions hold for tree CNs and, as formally shown below, are inherent to the survey propagation process.

**Assumption 1.** *For each clause $a$ and variable $i$, the random variables $M^s_{j \to a}$ for all $j \in V_i(a)$ are independent.*

**Assumption 2.** *For each clause $a$ and variable $i$, the random variables $M^u_{b \to i}$ for all clauses $b \in C^u_a(i)$ are independent.*

**Assumption 3.** *For each clause $a$ and variable $i$, the random variables $M^u_{b \to i}$ for all clauses $b \in C^s_a(i)$ are independent.*

Without any further assumptions, we can derive the following, by applying assumption 1 and the expression for $M^u_{a \to i}$ that results from 1:

$$p^u_{a \to i} = \mathbf{P}(M^u_{a \to i} = 1) = 1 - \prod_{j \in V_i(a)} \mathbf{P}(M^s_{j \to a} = 0) = 1 - \prod_{j \in V_i(a)} (1 - p^s_{j \to a}).$$

Similarly, by assumption 2 and the expression for $M^s_{i \to a}$ that results from 2, we derive

$$p^s_{i \to a} = \mathbf{P}(M^s_{i \to a} = 1) = \prod_{b \in C^u_a(i)} \mathbf{P}(M^u_{b \to i} = 1) = \prod_{b \in C^u_a(i)} p^u_{b \to i}.$$

Using the reparametrization $\eta_{a \to i} = \mathbf{P}(M^u_{a \to i} = 0) = 1 - p^u_{a \to i}$, we obtain the following message-passing procedure.

- *Message that clause $a$ sends to variable $i$:* $\eta_{a \to i} = \prod_{j \in V_i(a)} (1 - p^s_{i \to a})$
- *Message that variable $i$ sends to clause $a$:* $p^s_{i \to a} = \prod_{b \in C^u_a(i)} (1 - \eta_{b \to i})$

We can then use assumption 3 to estimate $p^u_{i \to a}$ as $\prod_{b \in C^s_a(i)} (1 - \eta_{b \to i})$.

Note that this message-passing procedure is exactly "classical" CP if we initialize $\eta_{a \to i} = 0$ and $p^s_{i \to a} = 1$ for all variables $i$ and clause $a$. However, the version here allows the messages to be in $[0, 1]$. At the same time, for tree CNs, this algorithm is the same as classical CP (*i.e.*, produces the same result), *regardless* of how the messages $\eta_{a \to i}$ and $p^s_{i \to a}$ are initialized. In fact, in the tree case, the final messages uniquely identify $\mathbf{P} = \mathbf{P}_f$.

**Making Assumptions about Satisfiability.** Let us make the following assumption about the "probabilistic satisfiability structure" of the $k$-SAT formula.

**Assumption 4.** *For some $\rho \in [0, 1]$, for each clause $a$ and variable $i$,*

$$\mathbf{P}(M^s_{i \to a} = 0, M^u_{i \to a} = 0) = (1 - \rho)\mathbf{P}(M^s_{i \to a} = 1, M^u_{i \to a} = 1).$$

For $\rho = 1$, the last assumption essentially says that $f_{a \to i}$ has a satisfying assignment; *i.e.*, $\mathbf{P}(M^s_{i \to a} = 0, M^u_{i \to a} = 0) = 0$. For $\rho = 0$, it essentially says that the likelihood that $f_{a \to i}$ *does not have* a satisfying assignment is the same as the likelihood that $f_{a \to i}$ *has* a satisfying assignment where variable $i$ is free. Formally, in this case, we have $\mathbf{P}(M^s_{i \to a} = 0, M^u_{i \to a} = 0) = \mathbf{P}(M^s_{i \to a} = 1, M^u_{i \to a} = 1)$, which, interestingly, is equivalent to the condition $\mathbf{P}(M^s_{i \to a} = 1) + \mathbf{P}(M^u_{i \to a} = 1) = 1$.

Let us introduce a final assumption about the random variables associated to the messages from variables to clauses.

**Assumption 5.** *For each clause $a$ and variable $i$, the random variables $M^s_{i \to a}$ and $M^u_{i \to a}$ are independent.*

Note that assumptions 2, 3 and 5 hold (simultaneously) if and only if for each clause $a$ and variable $i$, the random variables $M^u_{b \to i}$ for all clauses $b \in C_a(i)$ are independent.

The following theorem is the main result of this paper.

**Theorem 1. (Sufficient Assumptions)** *Let assumptions 1, 2 and 3 hold. The message-passing procedure that results from CPR as presented above is*

1. *belief propagation (i.e., $\mathrm{SP}(0)$), if assumption 4, with $\rho = 0$, holds, and*

2. *a member of the family of survey propagation algorithms $\mathrm{SP}(\rho)$, with $0 < \rho \leq 1$, if assumption 4, with the given $\rho$, and assumption 5 hold.*

These assumptions are also necessary in a strong sense (details omitted), Assumptions 1, 2, 3, and even 5 might be obvious to some readers, but assumption 4 might not be, and it is essential.

*Proof.* As in the last subsection, assumption 1 leads to $p^u_{a \to i} = 1 - \prod_{j \in V_i(a)} (1 - p^s_{j \to a})$, while assumptions 2 and 3 lead to $p^s_{i \to a} = \prod_{b \in C^u_a(i)} p^u_{b \to i}$ and $p^u_{i \to a} = \prod_{b \in C^s_a(i)} p^u_{b \to i}$.

Note also that assumption 4 is equivalent to $p^s_{i \to a} + p^u_{i \to a} - \rho\,\mathbf{P}(M^s_{i \to a} = 1, M^u_{i \to a} = 1) = 1$. This allows us to express

$$\mathbf{P}(M^s_{i \to a} = 1) = p^s_{i \to a} = \frac{p^s_{i \to a}}{p^s_{i \to a} + p^u_{i \to a} - \rho\,\mathbf{P}(M^s_{i \to a} = 1, M^u_{i \to a} = 1)},$$

which implies

$$\mathbf{P}(M^s_{i \to a} = 0) = \frac{p^u_{i \to a} - \rho\,\mathbf{P}(M^s_{i \to a} = 1, M^u_{i \to a} = 1)}{p^u_{i \to a} - \rho\,\mathbf{P}(M^s_{i \to a} = 1, M^u_{i \to a} = 1) + p^s_{i \to a}}.$$

If $\rho = 0$, then the last expression simplifies to

$$\mathbf{P}(M^s_{i \to a} = 0) = \frac{p^u_{i \to a}}{p^u_{i \to a} + p^s_{i \to a}}.$$

Using the reparametrization $\eta_{a\to i} \equiv \mathbf{P}(M^u_{a\to i} = 0) = 1 - p^u_{a\to i}$, $\Pi^u_{i\to a} \equiv \mathbf{P}(M^u_{i\to a} = 1) = p^u_{i\to a}$ and $\Pi^s_{i\to a} + \Pi^*_{i\to a} \equiv \mathbf{P}(M^s_{i\to a} = 1) = p^s_{i\to a}$, leads to BP (*i.e.*, SP(0)).

Otherwise, if $0 < \rho \le 1$, then using the reparametrization $\eta_{a\to i} \equiv \mathbf{P}(M^u_{a\to i} = 0)$,

$$
\begin{aligned}
\Pi^u_{i\to a} &\equiv \mathbf{P}(M^u_{i\to a} = 1) - \rho\,\mathbf{P}(M^s_{i\to a} = 1, M^u_{i\to a} = 1) \\
&= \mathbf{P}(M^s_{i\to a} = 0, M^u_{i\to a} = 1) + (1-\rho)\mathbf{P}(M^s_{i\to a} = 1, M^u_{i\to a} = 1), \\
\Pi^s_{i\to a} &\equiv \mathbf{P}(M^s_{i\to a} = 1, M^u_{i\to a} = 0), \text{ and} \\
\Pi^*_{i\to a} &\equiv \mathbf{P}(M^s_{i\to a} = 1, M^u_{i\to a} = 1),
\end{aligned}
$$

and applying assumption 5 leads to SP($\rho$). $\qquad\square$

The following are some remarks that can be easily derived using CPR.

**On the Relationship Between SP and BP.** SP essentially assumes that every sub-formula $f_{a\to i}$ has a satisfying assignment, while BP assumes that for every clause $a$ and variable $i \in V(a)$, variable $i$ is equally likely not to have a satisfying assignment or being free in $f_{a\to i}$, as it is easy to see from assumption 4. The parameter $\rho$ just modulates the relative scaling of those two likelihoods. While the same statement about pure SP is not novel, the statement about BP, and more generally, the class SP($\rho$) for $0 \le \rho < 1$, seems to be.

**On the Solutions of SAT formula $f$.** Note that $\mathbf{P}_f$ may not satisfy all or any of the assumptions. Yet, satisfying an assumption imposes constraints on what $\mathbf{P}_f$ actually is and thus on the solution space of $f$. For example, if $\mathbf{P}_f$ satisfies assumption 4 for any $\rho < 1$, which includes BP when $\rho = 0$, and for all clauses $a$ and variables $i$, then $\mathbf{P}_f(M^s_{i\to a} = 0, M^u_{i\to a} = 0) = \mathbf{P}_f(M^s_{i\to a} = 1, M^u_{i\to a} = 1) = 0$ and therefore either $\mathbf{P}_f(M^s_{i\to a} = 1, M^u_{i\to a} = 0) = 1$ or $\mathbf{P}_f(M^s_{i\to a} = 0, M^u_{i\to a} = 1) = 1$ holds, but not both of course. That implies $f$ must have a *unique* solution!

**On SP.** This result provides additional support to previous informal conjectures as to why SP is so effective near the satisfiability threshold: SP concentrates all its efforts on finding a satisfying assignment when they are scarce and "scattered" across the space of possible assignments. Thus, SP assumes that the set of satisfying assignments has in fact special structure.

To see that, note that assumptions 4, with $\rho = 1$, and 5 imply that $\mathbf{P}(M^s_{i\to a} = 1, M^u_{i\to a} = 0) = 0$ or $\mathbf{P}(M^s_{i\to a} = 0, M^u_{i\to a} = 1) = 0$ must hold. This says that in *every* assignment that satisfies $f_{a\to i}$, variable $i$ is either free or always has the same value assignment. This observation is relevant because it has been argued that as we approach the satisfiability threshold, the set of satisfying assignments decomposes into many "local" or disconnected subsets. It follows easily from the discussion here that SP assumes such a structure, therefore potentially making it most effective under those conditions (see Maneva et al. [3] for more information).

Similarly, it has also been empirically observed that SP is more effective for $\rho$ close to, but strictly less than 1. The CPR approach suggests that such behavior might be because, with respect to any $\mathbf{P}$ that satisfies assumption 4, unlike pure SP, for such values of $\rho < 1$, SP($\rho$) guards against the possibility that $f_{a\to i}$ is not satisfiable, while still being somewhat optimistic by giving more weight to the event that variable $i$ is free in $f_{a\to i}$. Naturally, BP, which is the case of $\rho = 0$, might be too pessimistic in this sense.

**On BP.** For BP ($\rho = 0$), making the additional assumption that the formula $f_{a\to i}$ is satisfiable (*i.e.*, $\mathbf{P}(M^s_{i\to a} = 0, M^u_{i\to a} = 0) = 0$) implies that there are no assignments with free variables (*i.e.*, $\mathbf{P}(M^s_{i\to a} = 1, M^u_{i\to a} = 1) = 0$). Therefore, the only possible consistent domain is the singleton $\{s_{a,i}\}$ or $\{u_{a,i}\}$ (*i.e.*, $\mathbf{P}(M^s_{i\to a} = 1, M^u_{i\to a} = 0) + \mathbf{P}(M^s_{i\to a} = 0, M^u_{i\to a} = 1) = 1$). Thus, either 0 or 1 can possibly be a consistent value assignment, but not both. This suggests that BP is concentrating its efforts on finding satisfying assignments *without* free variables.

**On Variable and Value Ordering.** To complete the picture of the derivation of SP($\rho$) via CPR, we need to compute $p^0_i$ and $p^1_i$ for all variables $i$ to use for variable and value ordering during search. We can use the following, slightly stronger versions of assumptions 2 and 3 for that.

**Assumption 6.** *For each variable $i$, the random variables $M^u_{a\to i}$ for all clauses $a \in C^-(i)$ are independent.*

**Assumption 7.** *For each variable $i$, the random variables $M_{a \to i}^u$ for all clauses $a \in C^+(i)$ are independent.*

Using assumptions 6 and 7, we can easily derive that $p_i^1 = \prod_{a \in C^-(i)}(1 - \eta_{a \to i})$ and $p_i^0 = \prod_{a \in C^+(i)}(1 - \eta_{a \to i})$, respectively.

**On Generalizations.** The approach provides a general, simple and principled way to introduce possibly uncertain domain knowledge into the problem by making assumptions about the structure of the set of satisfying assignments and incorporating them through $\mathbf{P}$. That can lead to more effective propagation algorithms for specific contexts.

**Related Work.** Dechter and Mateescu [10] also connect BP to CP but in the context of the inference problem of assessing zero posterior probabilities. Hsu and McIlraith [11] give an intuitive explanation of the behavior of SP and BP from the perspective of traditional local search methods. They provide a probabilistic interpretation, but the distribution used there is over the *biases*.

Braunstein and Zecchina [12] showed that pure SP is equivalent to BP on a particular MRF over an extended domain on the variables of the SAT formula, which adds a so called "joker" state. Maneva et al. [3] generalized that result by showing that $\text{SP}(\rho)$ is only one of many families of algorithms that are equivalent to performing BP on a particular MRF. In both cases, one can easily interpret those MRFs as ultimately imposing a distribution over $\Omega$, as defined here, where the joker state corresponds to the domain $\{0, 1\}$. Here, the only particular distribution explicitly defined is $\mathbf{P}_f$, the "optimal" distribution. This paper does not make any *explicit* statements about any *specific* distribution $\mathbf{P}$ for which applying CPR leads to $\text{SP}(\rho)$.

# 6 Conclusion

This paper strongly connects survey and constraint propagation. In fact, the paper shows how survey propagation algorithms are instances of CPR, the probabilistic generalization of classical constraint propagation proposed here. The general approach presented not only provides a new view on survey propagation algorithms, which can lead to a better understanding of them, but can also be used to easily develop potentially better algorithms tailored to specific classes of CSPs.

## Footnotes

[1] See also Pearl [4], section 4.1.1, and the first paragraph of section 4.1.2.

[2] Please refer to Russell and Norvig [6] for a general introduction, Kumar [7] for a tutorial and Dechter [8] for a more comprehensive treatment of these topics and additional references.

[3] Given clause $a$ and variable $i$ of SAT formula $f$, let $\mathcal{D}^j_{a \to i}$ be the (globally) consistent domain of $f_{a \to i}$ for variable $j$. The random variables corresponding to the messages from variable $i$ to clause $a$ are defined as $M^{x_i}_{i \to a}(\mathcal{W}) = 1$ iff $\mathcal{W}_j \subset \mathcal{D}^j_{a \to i}$ for every variable $j$ of $f_{a \to i}$; and $x_i \in \mathcal{D}^i_{a \to i}$. The other random variables are then defined as $M^s_{a \to i}(\mathcal{W}) = 1$ and $M^u_{a \to i}(\mathcal{W}) = 1 - \prod_{j \in V_i(a)} (1 - M^s_{j \to a}(\mathcal{W}))$ for all $\mathcal{W}$.

[4] For both variable and value ordering, we can break ties uniformly at random. Also, the description of $\text{SP}(\rho)$ used often, sets a *fraction* $\beta$ of the variables that remained unset during search. While clearly this speeds up the process of getting a full assignment, the effect that heuristic might have on the completeness of the search procedure is unclear, even in practice.

# References

[1] A. Braunstein, M. Mézard, and R. Zecchina. Survey propagation: An algorithm for satisfiability. *Random Structures and Algorithms*, 27:201, 2005.

[2] M. Mézard, G. Parisi, and R. Zecchina. Analytic and Algorithmic Solution of Random Satisfiability Problems. *Science*, 297(5582):812–815, 2002.

[3] E. Maneva, E. Mossel, and M. J. Wainwright. A new look at survey propagation and its generalizations. *ACM*, 54(4):2–41, July 2007.

[4] J. Pearl. *Probabilistic Reasoning in Intelligent Systems. Networks of Plausible Inference*. Morgan Kaufmann, 1988.

[5] D. L. Waltz. Generating semantic descriptions from drawings of scenes with shadows. Technical Report 271, MIT AI Lab, Nov. 1972. PhD Thesis.

[6] S. Russell and P. Norvig. *Artificial Intelligence: A Modern Approach*, chapter 5, pages 137–160. Prentice Hall, second edition, 1995.

[7] V. Kumar. Algorithms for constraint-satisfaction problems: A survey. *AI Magazine*, 13(1):32–44, 1992.

[8] R. Dechter. *Constraint Processing*. Morgan Kaufmann, 2003.

[9] F. Bacchus, X. Chen, P. van Beek, and T. Walsh. Binary vs. non-binary constraints. *AI*, 140(1-2):1–37, Sept. 2002.

[10] R. Dechter and R. Mateescu. A simple insight into iterative belief propagation's success. In *UAI*, 2003.

[11] E. I. Hsu and S. A. McIlraith. Characterizing propagation methods for boolean satisfiability. In *SAT*, 2006.

[12] A. Braunstein and R. Zecchina. Survey propagation as local equilibrium equations. *JSTAT*, 2004.

